# Detecting Significant Multidimensional Spatial Clusters

**Daniel B. Neill, Andrew W. Moore, Francisco Pereira, and Tom Mitchell**
School of Computer Science
Carnegie Mellon University
Pittsburgh, PA 15213
{neill,awm,fpereira,t.mitchell}@cs.cmu.edu

## Abstract

Assume a uniform, multidimensional grid of bivariate data, where each cell of the grid has a *count* $c_i$ and a *baseline* $b_i$. Our goal is to find spatial regions ($d$-dimensional rectangles) where the $c_i$ are significantly higher than expected given $b_i$. We focus on two applications: detection of clusters of disease cases from epidemiological data (emergency department visits, over-the-counter drug sales), and discovery of regions of increased brain activity corresponding to given cognitive tasks (from fMRI data). Each of these problems can be solved using a *spatial scan statistic* (Kulldorff, 1997), where we compute the maximum of a likelihood ratio statistic over all spatial regions, and find the significance of this region by randomization. However, computing the scan statistic for all spatial regions is generally computationally infeasible, so we introduce a novel *fast spatial scan* algorithm, generalizing the 2D scan algorithm of (Neill and Moore, 2004) to arbitrary dimensions. Our new multidimensional multiresolution algorithm allows us to find spatial clusters up to 1400x faster than the naive spatial scan, without any loss of accuracy.

## 1  Introduction

One of the core goals of modern statistical inference and data mining is to discover patterns and relationships in data. In many applications, however, it is important not only to discover patterns, but to distinguish those patterns that are *significant* from those that are likely to have occurred by chance. This is particularly important in epidemiological applications, where a rise in the number of disease cases in a region may or may not be indicative of an emerging epidemic. In order to decide whether further investigation is necessary, epidemiologists must know not only the location of a possible outbreak, but also some measure of the likelihood that an outbreak is occurring in that region. Similarly, when investigating brain imaging data, we want to not only find regions of increased activity, but determine whether these increases are significant or due to chance fluctuations.

More generally, we are interested in spatial data mining problems where the goal is detection of *overdensities*: spatial regions with high *counts* relative to some underlying *baseline*. In the epidemiological datasets, the count is some quantity (e.g. number of disease cases, or units of cough medication sold) in a given area, where the baseline is the expected value of that quantity based on historical data. In the brain imaging datasets, our count is the total fMRI activation in a given set of voxels under the experimental condition, while our baseline is the total activation in that set of voxels under the null or control condition.

We consider the case in which data has been aggregated to a uniform, $d$-dimensional grid. For the fMRI data, we have three spatial dimensions; for the epidemiological data, we have two spatial dimensions but also use several other quantities (time, patients' age and gender) as "pseudo-spatial" dimensions; this is discussed in more detail below.

In the general case, let $G$ be a $d$-dimensional grid of cells, with size $N_1 \times N_2 \times \ldots \times N_d$. Each cell $s_i \in G$ (where $i$ is a $d$-dimensional vector) is associated with a *count* $c_i$ and a *baseline* $b_i$. Our goal is to search over all $d$-dimensional rectangular regions $S \subseteq G$, and find regions where the total count $C(S) = \sum_S c_i$ is higher than expected, given the baseline $B(S) = \sum_S b_i$. In addition to discovering these high-density regions, we must also perform statistical testing to determine whether these regions are significant. As is necessary in the scan statistics framework, we focus on finding the single, most significant region; the method can be iterated (removing each significant cluster once it is found) to find multiple significant regions.

## 1.1 Likelihood ratio statistics

Our basic model assumes that counts $c_i$ are generated by an inhomogeneous Poisson process with mean $q b_i$, where $q$ (the underlying ratio of count to baseline) may vary spatially. We wish to detect hyper-rectangular regions $S$ such that $q$ is significantly higher inside $S$ than outside $S$. To do so, for a given region $S$, we assume that $q = q_{in}$ uniformly for cells $s_i \in S$, and $q = q_{out}$ uniformly for cells $s_i \in G - S$. We then test the null hypothesis $H_0(S)$: $q_{in} \leq (1 + \varepsilon) q_{out}$ against the alternative hypothesis $H_1(S)$: $q_{in} > (1 + \varepsilon) q_{out}$. If $\varepsilon = 0$, this is equivalent to the classical spatial scan statistic [1-2]: we are testing for regions where $q_{in}$ is greater than $q_{out}$. However, in many real-world applications (including the epidemiological and fMRI datasets discussed later) we expect some fluctuation in the underlying baseline; thus, we do not want to detect all deviations from baseline, but only those where the amount of deviation is greater than some threshold. For example, a 10% increase in disease cases in some region may not be interesting to epidemiologists, even if the underlying population is large enough to conclude that this is a "real" (statistically significant) increase in $q$. By increasing $\varepsilon$, we can focus the scan statistic on regions with larger ratios of count to baseline. For example, we can use the scan statistic with $\varepsilon = 0.25$ to test for regions where $q_{in}$ is more than 25% higher than $q_{out}$. Following Kulldorff [1], our spatial scan statistic is the maximum, over all regions $S$, of the ratio of the likelihoods under the alternative and null hypotheses. Taking logs for convenience, we have:

$$D_\varepsilon(S) = \log \frac{\sup_{q_{in} > (1+\varepsilon)q_{out}} \prod_{s_i \in S} P(c_i \sim \mathrm{Po}(q_{in} b_i)) \prod_{s_i \in G-S} P(c_i \sim \mathrm{Po}(q_{out} b_i))}{\sup_{q_{in} \leq (1+\varepsilon)q_{out}} \prod_{s_i \in S} P(c_i \sim \mathrm{Po}(q_{in} b_i)) \prod_{s_i \in G-S} P(c_i \sim \mathrm{Po}(q_{out} b_i))}$$

$$= (\mathrm{sgn}) \left( C(S) \log \frac{C(S)}{(1+\varepsilon)B(S)} + (C_{tot} - C(S)) \log \frac{C_{tot} - C(S)}{B_{tot} - B(S)} - C_{tot} \log \frac{C_{tot}}{B_{tot} + \varepsilon B(S)} \right)$$

where $C(S)$ and $B(S)$ are the count and baseline of the region $S$ under consideration, $C_{tot}$ and $B_{tot}$ are the total count and baseline of the entire grid $G$, and sgn = +1 if $\frac{C(S)}{B(S)} > (1 + \varepsilon) \frac{C_{tot} - C(S)}{B_{tot} - B(S)}$ and -1 otherwise. Then the scan statistic $D_{\varepsilon,max}$ is equal to the maximum $D_\varepsilon(S)$ over all spatial regions ($d$-dimensional rectangles) under consideration. We note that our statistical and computational methods are not limited to the Poisson model given here; any model of null and alternative hypotheses such that the resulting statistic $D(S)$ satisfies the conditions given in [4] can be used for the fast spatial scan.

## 1.2 Randomization testing

Once we have found the highest scoring region $S^* = \arg\max_S D(S)$ of grid $G$, we must still determine the statistical significance of this region. Since the exact distribution of the test statistic $D_{max}$ is only known in special cases, in general we must find the region's $p$-value by randomization. To do so, we run a large number $R$ of random replications, where a replica

has the same underlying baselines $b_i$ as $G$, but counts are randomly drawn from the null hypothesis $H_0(S^*)$. More precisely, we pick $c_i \sim \text{Po}(qb_i)$, where $q = q_{in} = (1+\varepsilon)\frac{C_{tot}}{B_{tot}+\varepsilon B(S^*)}$ for $s_i \in S^*$, and $q = q_{out} = \frac{C_{tot}}{B_{tot}+\varepsilon B(S^*)}$ for $s_i \in G - S^*$. The number of replicas $G'$ with $D_{max}(G') \geq D_{max}(G)$, divided by the total number of replications $R$, gives us the $p$-value for our most significant region $S^*$. If this $p$-value is less than $\alpha$ (where $\alpha$ is the false positive rate, typically chosen to be 0.05 or 0.1), we can conclude that the discovered region is statistically significant at level $\alpha$.

### 1.3 The naive spatial scan

The simplest method of finding $D_{max}$ is to compute $D(S)$ for all rectangular regions of sizes $k_1 \times k_2 \times \ldots \times k_d$, where $1 \leq k_j \leq N_j$. Since there are a total of $\prod_{j=1}^{d}(N_j - k_j + 1)$ regions of each size, there are a total of $O(\prod_{j=1}^{d} N_j^2)$ regions to examine. We can compute $D(S)$ for any region $S$ in constant time, by first finding the count $C(S)$ and baseline $B(S)$, then computing $D$.[1] This allows us to compute $D_{max}$ of a grid $G$ in $O(\prod_{j=1}^{d} N_j^2)$ time. However, significance testing by randomization also requires us to find $D_{max}$ for each replica $G'$, and compare this to $D_{max}(G)$; thus the total complexity is multiplied by the number of replications $R$. When the size of the grid is large, as is the case for the epidemiological and fMRI datasets we are considering, this naive approach is computationally infeasible.

Instead, we apply our "overlap-multiresolution partitioning" algorithm [3-4], generalizing this method from two-dimensional to $d$-dimensional datasets. This reduces the complexity to $O(\prod_{j=1}^{d} N_j \log N_j)$ in cases where the most significant region $S^*$ has a sufficiently high ratio of count to baseline, and (as we show in Section 3) typically results in tens to thousands of times speedup over the naive approach. We note that this *fast spatial scan* algorithm is exact (always finds the correct value of $D_{max}$ and the corresponding region $S^*$); the speedup results from the observation that we do not need to search a given set of regions if we can prove that none of them have score $> D_{max}$. Thus we use a top-down, *branch-and-bound* approach: we maintain the current maximum score of the regions we have searched so far, calculate upper bounds on the scores of subregions contained in a given region, and *prune* regions whose upper bounds are less than the current value of $D_{max}$. When searching a replica grid, we care only whether $D_{max}$ of the replica grid is greater than $D_{max}(G)$. Thus we can use $D_{max}$ of the original grid for pruning on the replicas, and can stop searching a replica if we find a region with score $> D_{max}(G)$.

## 2 Overlap-multiresolution partitioning

As in [4], we use a multiresolution search method which relies on an *overlap-kd tree* data structure. The overlap-kd tree, like kd-trees [5] and quadtrees [6], is a hierarchical, space-partitioning data structure. The root node of the tree represents the entire space under consideration (i.e. the entire grid $G$), and each other node represents a subregion of the grid. Each non-leaf node of a $d$-dimensional overlap-kd tree has $2d$ children, an "upper" and a "lower" child in each dimension. For example, in three dimensions, a node has six children: upper and lower children in the $x$, $y$, and $z$ dimensions. The overlap-kd tree is different from the standard kd-tree and quadtree in that adjacent regions overlap: rather than splitting the region in half along each dimension, instead each child contains *more* than half the area of the parent region. For example, a $64 \times 64 \times 64$ grid will have six children: two of size $48 \times 64 \times 64$, two of size $64 \times 48 \times 64$, and two of size $64 \times 64 \times 48$.

In general, let region $S$ have size $k_1 \times k_2 \times \ldots \times k_d$. Then the two children of $S$ in dimension $j$ (for $j = 1 \ldots d$) have size $k_1 \times \ldots \times k_{j-1} \times f_j k_j \times k_{j+1} \times \ldots \times k_d$, where $\frac{1}{2} < f_j < 1$. This partitioning (for the two-dimensional case, where $f_1 = f_2 = \frac{3}{4}$) is illustrated in Figure 1. Note that there is a region $S_C$ common to all of these children; we call this region the *center* of $S$. When we partition region $S$ in this manner, it can be proved that any subregion of $S$ either a) is contained entirely in (at least) one of $S_1 \ldots S_{2d}$, or b) contains the center region $S_C$. Figure 1 illustrates each of these possibilities, for the simple case of $d = 2$.

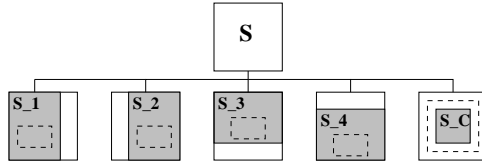

Figure 1: Overlap-multires partitioning of region $S$ (for $d = 2$). Any subregion of $S$ either a) is contained in some $S_i$, $i = 1 \ldots 4$, or b) contains $S_C$.

Now we can search all subregions of $S$ by recursively searching $S_1 \ldots S_{2d}$, then searching all of the regions contained in $S$ which contain the center $S_C$. There may be a large number of such "outer regions," but since we know that each such region contains the center, we can place very tight bounds on the score of these regions, often allowing us to prune most or all of them. Thus the basic outline of our search procedure (ignoring pruning, for the moment) is:

```
overlap-search(S)
{
  call base-case-search(S)
  define child regions S_1..S_2d, center S_C as above
  call overlap-search(S_i) for i=1..2d
  for all S' such that S' is contained in S and contains S_C, call base-case-search(S')
}
```

The fractions $f_i$ are selected based on the current sizes $k_i$ of the region being searched: if $k_i = 2^m$, then $f_i = \frac{3}{4}$, and if $k_i = 3 \times 2^m$, then $f_i = \frac{2}{3}$. For simplicity, we assume that all $N_i$ are powers of two, and thus all region sizes $k_i$ will fall into one of these two cases. Repeating this partitioning recursively, we obtain the overlap-kd tree structure. For $d = 2$, the first two levels of the overlap-kd tree are shown in Figure 2.

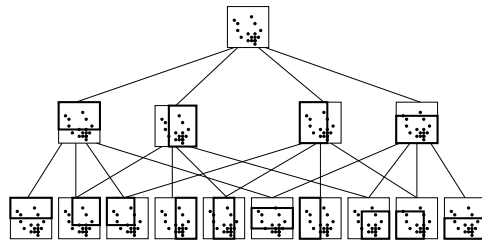

Figure 2: The first two levels of the two-dimensional overlap-kd tree. Each node represents a gridded region (denoted by a thick rectangle) of the entire dataset (thin square and dots).

The overlap-kd tree has several useful properties, which we present here without proof. First, for every rectangular region $S \subseteq G$, either $S$ is a gridded region (contained in the overlap-kd tree), or there exists a unique gridded region $S'$ such that $S$ is an outer region of $S'$ (i.e. $S$ is contained in $S'$, and contains the center region of $S'$). This means that, if overlap-search is called exactly once for each gridded region[2], and no pruning is done, then base-case-search will be called exactly once for every rectangular region $S \subseteq G$. In practice, we will prune many regions, so base-case-search will be called *at most once* for every rectangular region, and every region will be either searched or pruned. The second nice property of our overlap-kd tree is that the total number of gridded regions is $O(\prod_{j=1}^{d} N_j \log N_j)$. This implies that, if we are able to prune (almost) all outer regions, we can find $D_{max}$ of the grid in $O(\prod_{j=1}^{d} N_j \log N_j)$ time rather than $O(\prod_{j=1}^{d} N_j^2)$. In fact, we may not even need to search all gridded regions, so in many cases the search will be even faster.

[2] As in [4], we use "lazy expansion" to ensure that gridded regions are not multiply searched.

## 2.1 Score bounds and pruning

We now consider which regions can be *pruned* (discarded without searching) during our multiresolution search procedure. First, given some region $S$, we must calculate an upper bound on the scores $D(S')$ for regions $S' \subset S$. More precisely, we are interested in two upper bounds: a bound on the score of *all* subregions $S' \subset S$, and a bound on the score of the *outer* subregions of $S$ (those regions contained in $S$ and containing its center $S_C$). If the first bound is less than or equal to $D_{max}$, we can prune region $S$ completely; we do not need to search any (gridded or outer) subregion of $S$. If only the second bound is less than or equal to $D_{max}$, we do not need to search the outer subregions of $S$, but we must recursively call overlap-search on the gridded children of $S$. If both bounds are greater than $D_{max}$, we must both recursively call overlap-search and search the outer regions.

Score bounds are calculated based on various pieces of information about the subregions of $S$, including: upper and lower bounds $b_{max}$, $b_{min}$ on the baseline of subregions $S'$; an upper bound $d_{max}$ on the ratio $\frac{C}{B}$ of $S'$; an upper bound $d_{inc}$ on the ratio $\frac{C}{B}$ of $S' - S_C$; and a lower bound $d_{min}$ on the ratio $\frac{C}{B}$ of $S - S'$. We also know the count $C$ and baseline $B$ of region $S$, and the count $c_{center}$ and baseline $b_{center}$ of region $S_C$. Let $c_{in}$ and $b_{in}$ be the count and baseline of $S'$. To find an upper bound on $D(S')$, we must calculate the values of $c_{in}$ and $b_{in}$ which maximize $D$ subject to the given constraints: $\frac{c_{in} - c_{center}}{b_{in} - b_{center}} \leq d_{inc}$, $\frac{c_{in}}{b_{in}} \leq d_{max}$, $\frac{C - c_{in}}{B - b_{in}} \geq d_{min}$, and $b_{min} \leq b_{in} \leq b_{max}$. The solution to this maximization problem is derived in [4], and (since scores are based only on count and baseline rather than the size and shape of the region) it applies directly to the multidimensional case. The bounds on baselines and ratios $\frac{C}{B}$ are first calculated using global values (as a fast, "first-pass" pruning technique). For the remaining, unpruned regions, we calculate tighter bounds using the *quartering* method of [4], and use these to prune more regions.

## 2.2 Related work

Our work builds most directly on the results of Kulldorff [1], who presents the two-dimensional spatial scan framework and the classical ($\varepsilon = 0$) likelihood ratio statistic. It also extends [4], in which we present the two-dimensional fast spatial scan. Our major extensions in the present work are twofold: the $d$-dimensional fast spatial scan, and the generalized likelihood ratio statistics $D_{\varepsilon}$. A variety of other cluster detection techniques exist in the literature on epidemiology [1-3, 7-8], brain imaging [9-11], and machine learning [12-15]. The machine learning literature focuses on heuristic or approximate cluster-finding techniques, which typically cannot deal with spatially varying baselines, and more importantly, give no information about the statistical significance of the clusters found. Our technique is exact (in that it calculates the maximum of the likelihood ratio statistic over all hyper-rectangular spatial regions), and uses a powerful statistical test to determine significance. Nevertheless, other methods in the literature have some advantages over the present approach, such as applicability to high-dimensional data and fewer assumptions on the underlying model. The fMRI literature generally tests significance on a per-voxel basis (after applying some method of spatial smoothing); clusters must then be inferred by grouping individually significant voxels, and (with the exception of [10]) no per-cluster false positive rate is guaranteed. The epidemiological literature focuses on detecting significant circular, two-dimensional clusters, and thus cannot deal with multidimensional data or elongated regions. Detection of elongated regions is extremely important in both epidemiology (because of the need to detect windborne or waterborne pathogens) and brain imaging (because of the "folded sheet" structure of the brain); the present work, as well as [4], allow detection of such clusters.

# 3 Results

We now describe results of our fast spatial scan algorithm on three sets of real-world data: two sets of epidemiological data (from emergency department visits and over-the-counter

drug sales), and one set of fMRI data. Before presenting these results, we wish to emphasize three main points. First, the extension of scan statistics from two-dimensional to $d$-dimensional datasets dramatically increases the scope of problems for which these techniques can be used. In addition to datasets with more than two spatial dimensions (for example, the fMRI data, which consists of a 3D picture of the brain), we can also examine data with a temporal component (as in the OTC dataset), or where we wish to take demographic information into account (as in the ED dataset). Second, in all of these datasets, the use of the broader class of likelihood ratio statistics $D_\varepsilon$ (instead of only the classical scan statistic $\varepsilon = 0$) allows us to focus our search on smaller, denser regions rather than slight (but statistically significant) increases over a large area. Third, as our results here will demonstrate, the fast spatial scan gains huge performance improvements over the naive approach, making the use of the scan statistic feasible in these large, real-world datasets.

Our first test set was a database of (anonymized) Emergency Department data collected from Western Pennsylvania hospitals in the period 1999-2002. This dataset contains a total of 630,000 records, each representing a single ED visit and giving the latitude and longitude of the patient's home location to the nearest $\frac{1}{3}$ mile (a sufficiently low resolution to ensure anonymity). Additionally, a record contains information about the patient's gender and age decile. Thus we map records into a four-dimensional grid, consisting of two spatial dimensions (longitude, latitude) and two "pseudo-spatial" dimensions (patient gender and age decile). This has several advantages over the traditional (two-dimensional) spatial scan. First, our test has higher power to detect syndromes which affect differing patient demographics to different extents. For example, if a disease primarily strikes male infants, we might find a cluster with gender = male and age decile = 0 in some spatial region, and this cluster may not be detectable from the combined data. Second, our method accounts correctly for multiple hypothesis testing. If we were to instead perform a separate test at level $\alpha$ on each combination of gender and age decile, the overall false positive rate would be much higher than $\alpha$. We mapped the ED dataset to a $128 \times 128 \times 2 \times 8$ grid, with the first two coordinates corresponding to longitude and latitude, the third coordinate corresponding to the patient's gender, and the fourth coordinate corresponding to the patient's age decile. We tested for spatial clustering of "recent" disease cases: the count of a cell was the number of ED visits in that spatial region, for patients of that age and gender, in 2002, and the baseline was the total number of ED visits in that spatial region, for patients of that age and gender, over the entire temporal period 1999-2002. We used the $D_\varepsilon$ scan statistic with values of $\varepsilon$ ranging from 0 to 1.0. For the classical scan statistic ($\varepsilon = 0$), we found a region of size $35 \times 34 \times 2 \times 8$; thus the most significant region was spatially localized but cut across all genders and age groups. The region had $C = 3570$ and $B = 6409$, as compared to $\frac{C}{B} = 0.05$ outside the region, and thus this is clearly an overdensity. This was confirmed by the algorithm, which found the region statistically significant ($p$-value 0/100). With the three other values of $\varepsilon$, the algorithm found almost the same region ($35 \times 33 \times 2 \times 8$, $C = 3566$, $B = 6390$) and again found it statistically significant ($p$-value 0/100). For all values of $\varepsilon$, the fast scan statistic found the most significant region hundreds of times faster than the naive spatial scan (see Table 1): while the naive approach required approximately 12 hours per replication, the fast scan searched each replica in approximately 2 minutes, plus 100 minutes to search the original grid. Thus the fast algorithm achieved speedups of 235-325x over the naive approach for the entire run (i.e. searching the original grid and 100 replicas) on the ED dataset.

Our second test set was a nationwide database of retail sales of over-the-counter cough and cold medication. Sales figures were reported by zip code; the data covered 5000 zip codes across the U.S. In this case, our goal was to see if the spatial distribution of sales in a given week (February 7-14, 2004) was significantly different than the spatial distribution of sales during the previous week, and to identify a significant cluster of increased sales if one exists. Since we wanted to detect clusters even if they were only present for part of the week, we used the date (Feb. 7-14) as a third dimension. This is similar to the retrospective

Table 1: Performance of algorithm, real-world datasets

| test | ε | sec/orig | sec/rep | speedup | regions (orig) | regions (rep) |
|---|---|---|---|---|---|---|
| ED | 0 | 6140 | 126 | x235 | 358M | 622K |
| $(128 \times 128 \times 2 \times 8)$ | 0.25 | 6035 | 100 | x275 | 352M | 339K |
| (7.35B regions) | 0.5 | 5994 | 102 | x272 | 348M | 362K |
| | 1.0 | 5607 | 79.6 | x325 | 334M | 336K |
| OTC | 0 | 4453 | 195 | x48 | 302M | 2.46M |
| $(128 \times 128 \times 8)$ | 0.25 | 429 | 123 | x90 | 12.2M | 1.39M |
| (2.45B regions) | 0.5 | 334 | 51 | x210 | 8.65M | 350K |
| | 1.0 | 229 | 5.9 | x1400 | 4.40M | < 10 |
| fMRI | 0 | 880 | 384 | x7 | 39.9M | 14.0M |
| $(64 \times 64 \times 16)$ | 0.01 | 597 | 285 | x9 | 35.2M | 10.4M |
| (588M regions) | 0.02 | 558 | 188 | x14 | 33.1M | 6.65M |
| | 0.03 | 547 | 97.3 | x27 | 32.3M | 3.93M |
| | 0.04 | 538 | 30.0 | x77 | 31.9M | 1.44M |
| | 0.05 | 538 | 13.1 | x148 | 31.7M | 310K |

space-time scan statistic of [16], which also uses time as a third dimension. However, that algorithm searches over cylinders rather than hyper-rectangles, and thus cannot detect spatially elongated clusters. The count of a cell was taken to be the number of sales in that spatial region on that day; to adjust for day-of-week effects, the baseline of a cell was taken to be the number of sales in that spatial region on the day one week prior (Jan. 31-Feb. 7). Thus we created a $128 \times 128 \times 8$ grid, where the first two coordinates were derived from the longitude and latitude of that zip code, and the third coordinate was temporal, based on the date. For this dataset, the classical scan statistic ($\varepsilon = 0$) found a region of size $123 \times 76$ from February 7-11. Unfortunately, since the ratio $\frac{C}{B}$ was only 0.99 inside the region (as compared to 0.96 outside) this region would not be interesting to an epidemiologist. Nevertheless, the region was found to be significant ($p$-value 0/100) because of the large total baseline. Thus, in this case, the classical scan statistic finds a large region of very slight overdensity rather than a smaller, denser region, and thus is not as useful for detecting epidemics. For $\varepsilon = 0.25$ and $\varepsilon = 0.5$, the scan statistic found a much more interesting region: a $4 \times 1$ region on February 9 where $C = 882$ and $B = 240$. In this region, the number of sales of cough medication was 3.7x its expected value; the region's $p$-value was computed to be 0/100, so this is a significant overdensity. For $\varepsilon = 1$, the region found was almost the same, consisting of three of these four cells, with $C = 825$ and $B = 190$. Again the region was found to be significant ($p$-value 0/100). For this dataset, the naive approach took approximately three hours per replication. The fast scan statistic took between six seconds and four minutes per replication, plus ten minutes to search the original grid, thus obtaining speedups of 48-1400x on the OTC dataset.

Our third and final test set was a set of fMRI data, consisting of two "snapshots" of a subject's brain under null and experimental conditions respectively. The experimental condition was from a test [9] where the subject is given words, one at a time; he must read these words and identify them as verbs or nouns. The null condition is the subject's average brain activity while fixating on a cursor, before any words are presented. Each snapshot consists of a $64 \times 64 \times 16$ grid of voxels, with a reading of fMRI activation for the subset of the voxels where brain activity is occurring. In this case, the count of a cell is the fMRI activation for that voxel under the experimental condition, and the baseline is the activation for that voxel under the null condition. For voxels with no brain activity, we have $c_i = b_i = 0$. For the fMRI dataset, the amount of change between activated and non-activated regions is small, and thus we used values of $\varepsilon$ ranging from 0 to 0.05.

For the classical scan statistic ($\varepsilon = 0$) our algorithm found a $23 \times 20 \times 11$ region, and again found this region significant ($p$-value 0/100). However, this is another example where the

classical scan statistic finds a region which is large ($\frac{1}{4}$ of the entire brain) and only slightly increased in count: $\frac{C}{B} = 1.007$ inside the region and $\frac{C}{B} = 1.002$ outside the region. For $\varepsilon = 0.01$, we find a more interesting cluster: a $5 \times 10 \times 1$ region in the visual cortex containing four non-zero voxels.[3] For this region $\frac{C}{B} = 1.052$, a large increase, and the region is significant at $\alpha = 0.1$ ($p$-value 10/100) though not at $\alpha = 0.05$. For $\varepsilon = 0.02$, we find the same region, but conclude that it is not significant ($p$-value 32/100). For $\varepsilon = 0.03$ and $\varepsilon = 0.04$, we find a $3 \times 2 \times 1$ region with $\frac{C}{B} = 1.065$, but this region is not significant ($p$-values 61/100 and 89/100 respectively). Similarly, for $\varepsilon = 0.05$, we find a single voxel with $\frac{C}{B} = 1.075$, but again it is not significant ($p$-value 91/100). For this dataset, the naive approach took approximately 45 minutes per replication. The fast scan statistic took between 13 seconds and six minutes per replication, thus obtaining speedups of 7-148x on the fMRI dataset.

Thus we have demonstrated (through tests on a variety of real-world datasets) that the fast multidimensional spatial scan statistic has significant performance advantages over the naive approach, resulting in speedups up to 1400x without any loss of accuracy. This makes it feasible to apply scan statistics in a variety of application domains, including the spatial and spatio-temporal detection of disease epidemics (taking demographic information into account), as well as the detection of regions of increased brain activity in fMRI data. We are currently examining each of these application domains in more detail, and investigating which statistics are most useful for each domain. The generalized likelihood ratio statistics presented here are a first step toward this: by adjusting the parameter $\varepsilon$, we can "tune" the statistic to detect smaller and denser, or larger but less dense, regions as desired, and our statistical significance test is adjusted accordingly. We believe that the combination of fast computational algorithms and more powerful statistical tests presented here will enable the multidimensional spatial scan statistic to be useful in these and many other applications.

## Footnotes

[1] An old trick makes it possible to compute the count and baseline of any rectangular region in time constant in $N$: we first form a $d$-dimensional array of the cumulative counts, then compute each region's count by adding/subtracting at most $2^d$ cumulative counts. Note that because of the exponential dependence on $d$, these techniques suffer from the "curse of dimensionality": neither the naive spatial scan, nor the fast spatial scan discussed below, are appropriate for very high dimensional datasets.

[3]In a longer run on a different subject, where we iterate the scan statistic to pick out multiple significant regions, we found significant clusters in Broca's and Wernicke's areas in addition to the visual cortex. This makes sense given the nature of the experimental task; however, more data is needed before we can draw conclusive cross-subject comparisons.

## References

[1] M. Kulldorff. 1997. A spatial scan statistic. *Communications in Statistics: Theory and Methods* **26**(6), 1481-1496.

[2] M. Kulldorff. 1999. Spatial scan statistics: models, calculations, and applications. In Glaz and Balakrishnan, eds. *Scan Statistics and Applications*. Birkhauser: Boston, 303-322.

[3] D. B. Neill and A. W. Moore. 2003. A fast multi-resolution method for detection of significant spatial disease clusters. In *Advances in Neural Information Processing Systems* **16**.

[4] D. B. Neill and A. W. Moore. 2004. Rapid detection of significant spatial clusters. To be published in *Proc. 10th ACM SIGKDD Intl. Conf. on Knowledge Discovery and Data Mining*.

[5] J. L. Bentley. 1975. Multidimensional binary search trees used for associative searching. *Comm. ACM* **18**, 509-517.

[6] R. A. Finkel and J. L. Bentley. 1974. Quadtrees: a data structure for retrieval on composite keys. *Acta Informatica* **4**, 1-9.

[7] S. Openshaw, et al. 1988. Investigation of leukemia clusters by use of a geographical analysis machine. *Lancet* **1**, 272-273.

[8] L. A. Waller, et al. 1994. Spatial analysis to detect disease clusters. In N. Lange, ed. *Case Studies in Biometry*. Wiley, 3-23.

[9] T. Mitchell et al. 2003. Learning to detect cognitive states from brain images. *Machine Learning*, in press.

[10] M. Perone Pacifico et al. 2003. False discovery rates for random fields. Carnegie Mellon University Dept. of Statistics, Technical Report 771.

[11] K. Worsley et al. 2003. Detecting activation in fMRI data. *Stat. Meth. in Medical Research* **12**, 401-418.

[12] R. Agrawal, et al. 1998. Automatic subspace clustering of high dimensional data for data mining applications. *Proc. ACM-SIGMOD Intl. Conference on Management of Data*, 94-105.

[13] J. H. Friedman and N. I. Fisher. 1999. Bump hunting in high dimensional data. *Statistics and Computing* **9**, 123-143.

[14] S. Goil, et al. 1999. MAFIA: efficient and scalable subspace clustering for very large data sets. *Northwestern University, Technical Report CPDC-TR-9906-010*.

[15] W. Wang, et al. 1997. STING: a statistical information grid approach to spatial data mining. *Proc. 23rd Conference on Very Large Databases*, 186-195.

[16] M. Kulldorff. 1998. Evaluating cluster alarms: a space-time scan statistic and brain cancer in Los Alamos. *Am. J. Public Health* **88**, 1377-1380.

